# Text-Based Information Retrieval Using Exponentiated Gradient Descent

**Ron Papka, James P. Callan, and Andrew G. Barto** *
Department of Computer Science
University of Massachusetts
Amherst, MA 01003
papka@cs.umass.edu, callan@cs.umass.edu, barto@cs.umass.edu

## Abstract

The following investigates the use of single-neuron learning algorithms to improve the performance of text-retrieval systems that accept natural-language queries. A retrieval process is explained that transforms the natural-language query into the query syntax of a real retrieval system: the initial query is expanded using statistical and learning techniques and is then used for document ranking and binary classification. The results of experiments suggest that Kivinen and Warmuth's Exponentiated Gradient Descent learning algorithm works significantly better than previous approaches.

## 1  Introduction

The following work explores two learning algorithms – Least Mean Squared (LMS) [1] and Exponentiated Gradient Descent (EG) [2] – in the context of text-based Information Retrieval (IR) systems. The experiments presented in [3] use connectionist learning models to improve the retrieval of relevant documents from a large collection of text. Here, we present further analysis of those experiments. Previous work in the area employs various techniques for improving retrieval [6, 7, 14]. The experiments presented here show that EG works significantly better than widely used ad hoc methods for finding a good set of query term weights.

The retrieval processes being considered operate on a collection of documents, a natural-language query, and a training set of documents judged relevant or non-relevant to the query. The query may be, for example, the information request submitted through a web-search engine, or through the interface of a system with

---

This material is based on work supported by the National Science Foundation, Library of Congress, and Department of Commerce under cooperative agreement number EEC-9209623. Any opinions, findings and conclusions or recommendations expressed in this material are those of the author and do not necessarily reflect those of the sponsor.

domain-specific information such as legal, governmental, or news data maintained as a collection of text. The query, expressed as complete or incomplete sentences, is modified through a learning process that incorporates the terms in the test collection that are important for improving retrieval performance. The resulting query can then be used against collections similar in domain to the training collection.

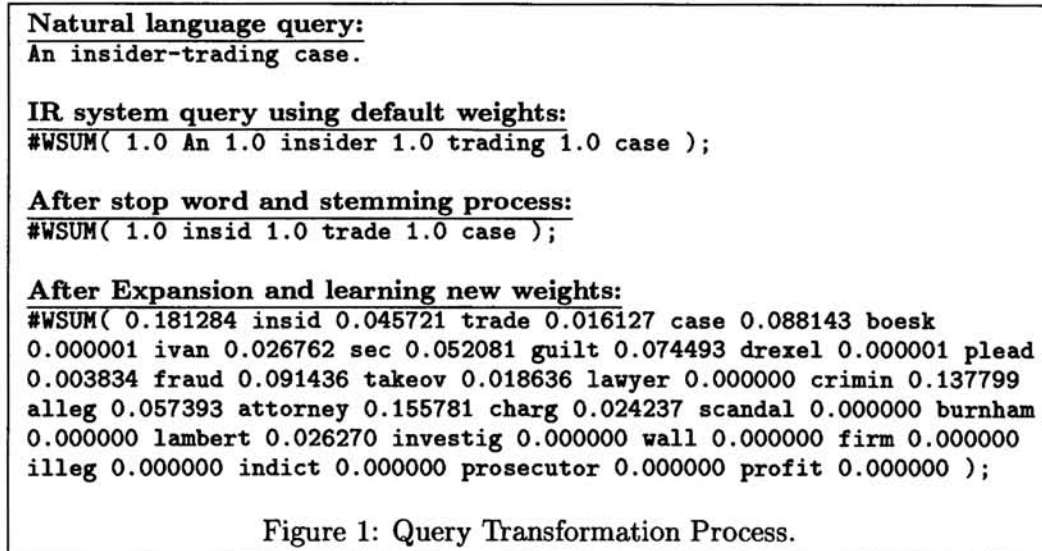

Figure 1: Query Transformation Process.

The query transformation process is illustrated in Figure 1. First, the natural-language query is transformed into one which can be used by the query-parsing mechanism of the IR system. The weights associated with each term are assigned a default value of 1.0, implying that each term is equally important in discriminating relevant documents. The query then undergoes a *stopping and stemming* process, by which morphological stemming and the elimination of very common words, called stopwords, increases both the effectiveness and efficiency of a system [9]. The query is subsequently expanded using a statistical term-expansion process producing terms from the training set of documents. Finally, a learning algorithm is invoked to produce new weights for the expanded query.

## 2  Retrieval Process

Text-based information retrieval systems allow the user to pose a query to a collection or a stream of documents. When a query $q$ is presented to a collection $c$, each document $d \in c$ is examined and assigned a value relative to how well $d$ satisfies the semantics of the request posed by $q$. For any instance of the triple $< q, d, c >$, the system determines an evaluation value attributed to $d$ using the function $eval(q, d, c)$.

The evaluation function $eval(q, d, c) = \dfrac{\sum_{i=1}^{N} q_i * d_i}{\sum_{i=1}^{N} q_i}$

was used for this work, and is based on an implementation of INQUERY [8]. It is assumed that $q$ and $d$ are vectors of real numbers, and that $c$ contains precomputed collection statistics in addition to the current set of documents. Since the collection may change over time, it may be necessary to change the query representation over time; however, in what follows the training collection is assumed to be static, and successful learning implies that the resulting query generalizes to similar collections.

An IR system can perform several kinds of retrieval tasks. This work is specifically concerned with two retrieval processes: document ranking and document classification. A ranking of documents based on query $q$ is achieved by sorting all documents in a collection by evaluation value. Binary classification is achieved by determining a threshold $\theta$ such that for class $R$, $eval(q, d, c) \geq \theta \rightarrow d \in R$, and

$eval(q, d, c) < \theta \rightarrow d \in \bar{R}$, so that $R$ is the set of documents from the collection that are classified as relevant to the query, and $\bar{R}$ is the set classified as non-relevant.

Central to any IR system is a parsing process used for documents and queries, which produces tokens called *terms*. The terms derived from a document are used to build an *inverted list* structure which serves as an index to the collection. The natural-language query is also parsed into a set of terms. Research-based IR systems such as INQUERY, OKAPI [11], and SMART [5], assume that the co-occurrence of a term in a query and a document indicates that the document is relevant to the query to some degree, and that a query with multiple terms requires a mechanism by which to combine the evidence each co-occurrence contributes to the document's degree of relevance to the query. The document representation for such systems is a vector, each element of which is associated with a unique term in the document. The values in the vector are produced by a *term-evaluation function* comprised of a term frequency component, $tf$, and an inverse document frequency component, $idf$, which are described in [8, 11]. The $tf$ component causes the term-evaluation value to increase as a query-term's occurrence in the document increases, and the $idf$ component causes the term-evaluation value to decrease as the number of documents in the collection in which the term occurs increases.

## 3 Query Expansion

Though it is possible to learn weights for terms in the original query, better results are obtained by first expanding the query with additional terms that can contribute to identifying relevant documents, and then learning the weights for the expanded query. The optimal number of terms by which to expand a query is domain-dependent, and query expansion can be performed using several techniques, including thesaurus expansion and statistical methods [12]. The query expansion process performed in this work is a two-step process: *term selection* followed by *weight assignment*. The term selection process ranks all terms found in relevant documents by an information metric described in [8]. The top $n$ terms are used in the expanded query. The experiments in this work used values of 50 and 1000 for $n$. The most common technique for weight assignment is derived from a closed-form function originally presented by Rocchio in [6], but our experiments show that a single-neuron learning approach is more effective.

### 3.1 Rocchio Weights

We assume that the terms of the original query are stored in a vector $t$, and that their associated weights are stored in $q$. Assuming that the new terms in the expanded query are stored $t'$, the weights for $q'$ can be determined using a method originally developed by Rocchio that has been improved upon in [7, 8]. Using the notation presented above, the weight assignment can be represented in the linear form: $q' = \alpha * f(t) + \beta * r(t', R_q, c) + \gamma * nr(t', \bar{R}_q, c)$, where $f$ is a function operating on the terms in the original query, $r$ is a function operating on the term statistics available from the training set of relevant documents ($R_q$), and $nr$ is a function operating on the statistics from the non-relevant documents ($\bar{R}_q$). The values for $\alpha$, $\beta$, and $\gamma$ have been the focus of many IR experiments, and 1.0, 2.0, and 0.5, have been found to work well with various implementations of the functions $f$, $r$, and $nr$ [7].

### 3.2 LMS and EG

In the experiments that follow, LMS and EG were used to learn query term weights. Both algorithms were used in a training process attempting to learn the association between the set of training instances (documents) and their corresponding binary classifications (relevant or non-relevant). A set of weights $\vec{w}$ is updated given an input instance $\vec{x}$ and a target binary classification value $y$. The algorithms learn the association between $\vec{x}$ and $y$ perfectly if $\vec{w} \cdot \vec{x} = y$, otherwise the value $(y - \vec{w} \cdot \vec{x})$ is the error or *loss* incurred. The task of the learning algorithm is to learn the values of $\vec{w}$ for more than one instance of $\vec{x}$.

The update rule for LMS is $\vec{w}_{t+1} = \vec{w}_t + \vec{r}_t$, where $\vec{r}_t = -2\eta_t(\vec{w}_t \cdot \vec{x}_t - y_t)\vec{x}_t$, where the step-size $\eta_t = \frac{1}{\vec{x}_t \cdot \vec{x}_t}$. The update rule for EG is $\vec{w}_{t+1,i} = \frac{\vec{w}_{t,i} e^{\vec{r}_{t,i}}}{\sum_{j=1}^{N} \vec{w}_{t,j} e^{\vec{r}_{t,j}}}$, where

$\vec{r}_{t,i} = -2\eta_t(\vec{w}_t \cdot \vec{x}_t - y_t)\vec{x}_{t,i}$, and $\eta_t = \frac{2}{3(\max_i(\vec{x}_{t,i}) - \min_i(\vec{x}_{t,i}))}$.

There are several fundamental differences between LMS and EG; the most salient is that EG has a multiplicative exponential update rule, while LMS is additive. A less obvious difference is the derivation of these two update rules. Kivinen and Warmuth [2] show that both rules are *approximately* derivable from an optimization task that minimizes the linear combination of a distance and a loss function: $distance(\vec{w}_{t+1}, \vec{w}_t) + \eta_t loss(y_t, \vec{w}_t \cdot \vec{x}_t)$. But the *distance* component for the derivation leading to the LMS update rule uses the squared Euclidean distance $\|\vec{w}_{t+1} - \vec{w}_t\|_2^2$, while the derivation leading to the EG update rule uses relative entropy or $\sum_{i=1}^N \vec{w}_{t+1,i} \ln \frac{\vec{w}_{t+1,i}}{\vec{w}_{t,i}}$. Entropy metrics had previously been used as the *loss* component [4].

One purpose of Kivinen and Warmuth's work was to describe loss bounds for these algorithms; however, they also observed that EG suffers significantly less from irrelevant attributes than does LMS. This hypothesis was tested in the experiments conducted for this work.

## 4  Experiments

Experiments were conducted on 100 natural-language queries. The queries were manually transformed into INQUERY syntax, expanded using a statistical technique described in [8], and then given a weight assignment as a result of a learning process. One set of experiments expanded each query by 50 terms and another set of experiments expanded each query by 1000 terms. The purpose of the latter was to test the ability of each algorithm to learn in the presence of many irrelevant attributes.

### 4.1  Data

The queries used are the *description* fields of information requests developed for Text Retrieval Conferences (TREC) [10]. The first set of queries was taken from TREC topics 51-100 and the second set from topics 101-150, for a total of 100 queries. After stopping and stemming, the average number of terms remaining before expansion was 8.34 terms.

Training and testing for all queries was conducted on subsets of the Tipster collection, which currently contains 3.4 gigabytes of text, including 206,201 documents whose relevance to the TREC topics has been evaluated. The collection is partitioned into 3 volumes. The judged documents from volumes 1 and 2 were used for training, while the documents from volume 3 were used for testing. Volumes 1 and 2 contain 741,856 documents from the Associated Press(1988-9), Department of Energy abstract, Federal Register(1988-9), Wall Street Journal(1987-91), and Ziff-Davis Computer-select articles. Volume 3 contains 336,310 documents from Associated Press(1990), San Jose Mercury News(1991), and Ziff-Davis articles.

Only a subset of the data for the TREC-Tipster environment has been judged. Binary judgments are assessed by humans for the top few thousand documents that were retrieved for each query by participating systems from various commercial and research institutions. Based on the judged documents available for volumes 1 and 2, on average 280 relevant documents and 1236 non-relevant documents were used to train each query.

### 4.2  Training Parameters

Rocchio weights were assigned based on coefficients described in Section 3.1. LMS and EG update rules were applied using 100,000 random presentations of training instances. It was empirically determined that this number of presentations was sufficient to allow both learning algorithms to produce better query weights than the Rocchio assignment based on performance metrics calculated using the training instances.

In reality, of course, the number of documents that will be relevant to a particular query is much smaller than the number of documents that are non-relevant. This property gives rise to the question of what is an appropriate sampling bias

of training instances, considering that the ratio of relevant to non-relevant documents approaches 0 in the limit. In the following experiments, LMS benefitted from uniform random sampling from the set of training instances, while EG benefitted from a balanced sampling, that is uniform random sampling from relevant training instances on even iterations and from non-relevant instances on odd iterations.

A *pocketing technique* was applied to the learning algorithms [13]. The purpose of this technique is to find a set of weights that optimize a specific user's utility function. In the following experiments, weights were tested every 1000 iterations using a recall and precision performance metric. If a set of weights produced a new performance-metric maximum, it was saved. The last set saved was assumed to be the result of the algorithm, and was used for testing.

A binary classification value pair $(A, B)$ is supplied as the target for training, where $A$ is the classification value for relevant documents, and $B$ is the classification value for non-relevant documents. Using the standard classification value pair (1, 0), INQUERY's document representation inhibits learning due to the large error caused by these unattainable values. Therefore, testing was done and resulted in the observation that .4 was the lowest attainable evaluation value for a document, and .47 appeared to be a good classification value for relevant documents. The classification value pair used for both the LMS and EG algorithms was thus (.47, .40).

## 4.3 Evaluation

In the experiments that follow, R-Precision (RP) was used to evaluate ranking performance, and a new metric, Lower Bound Accuracy (LBA) was used to evaluate classification. Both metrics make use of *recall* and *precision*, which are defined as follows: Assume there exists a set of documents sorted by evaluation value and a process that has performed classification, and that $a$ = number of relevant documents classified as relevant, $b$ = number of non-relevant documents classified as relevant, $c$ = number of relevant documents classified as non-relevant, and $d$ = number of non-relevant documents classified as non-relevant; then, $Recall = \frac{a}{a+c}$, and $Precision = \frac{a}{a+b}$ [3].

*Precision* and *recall* can be calculated at any cut-off point in the sorted list of documents. *R-Precision* is calculated using the top $n$ documents, where $n$ is the number of relevant training documents available for a query.

*Lower Bound Accuracy* (LBA) is a metric that assumes the minimum of a classifier's accuracy with respect to relevant documents and its accuracy with respect to non-relevant documents. It is defined as $LBA = min(\frac{a}{a+c}, \frac{d}{b+d})$. An LBA value can be interpreted as the lower bound of the percent of instances a classifier will correctly classify, regardless of an imbalance between the actual number of relevant and non-relevant documents. This metric requires a threshold $\theta$. The threshold is taken to be the evaluation value of the document at a cut-off point in the sorted list of training documents where LBA is maximized. Hence, $\theta = max_i(LBA(d_i, R_q, \bar{R}_q))$, where $d_i$ is the $i$th document in the sorted list.

## 4.4 Results

| Query type | RP | LBA |
|------------|------|------|
| NL | 22.0 | 88.6 |
| EXP | 28.7 | 92.0 |
| ROC | 33.4 | 94.0 |
| LMS | 32.5 | 89.8 |
| EG | 40.3 | 95.1 |

Table 1: Query expansion by 50 terms

The following results show the ability of a query weight assignment to generalize. The weights are derived from a subset of the training collection, and the values reported are based on performance on the test collection. The results of the 50-term-expansion experiments are listed in Table 1 [1]. They indicate that the expanded query has an advantage over the original query, and that the EG-trained query generalized better than the other algorithms, while Rocchio appears to be the next best. In terms of ranking, EG gives rise to a 20% improvement over the Rocchio assignment, and realizes 1.2% improvement in terms of classification. This apparently slight improvement in classification in fact implies that EG is correctly classifying at least 3000 documents more than the other approaches.

Table 2 shows a cross-algorithm analysis in which any two algorithms can be compared. The analysis is calculated using both RP and LBA over all queries. An entry for row $i$ column $j$ indicates the number of queries for which the performance of algorithm $i$ was better than algorithm $j$. Based on sign tests with $\alpha = .01$, the results confirm that EG significantly generalized better than the other algorithms.[2]

| Query type | Query counts: RP-LBA | | | | |
|---|---|---|---|---|---|
|  | NL | EXP | ROC | LMS | EG |
| NL | - | 30 -37 | 18 - 13 | 24 - 53 | 12 - 13 |
| EXP | 60 - 62 | - | 9 - 17 | 35 - 66 | 11 - 19 |
| ROC | 71 - 86 | 72 -79 | - | 53 - 73 | 17 - 37 |
| LMS | 66 - 46 | 54 -34 | 38 - 26 | - | 13 - 15 |
| EG | 79 - 85 | 80 -80 | 70 - 62 | 74 - 84 | - |

Table 2: Cross Algorithm Analysis over 100 queries expanded by 50 terms.

As explained in Section 4.3, the thresholds used to calculate the LBA performance metric are determined by obtaining an evaluation value in the training data corresponding to the cut-off point where LBA was maximized. The threshold analysis in Table 3 shows the best attainable classification performance against performance actually achieved. The results indicate that there is still room for improvement; however, they also indicate that this methodology is acceptable.

The results for queries expanded by 1000 terms are listed in Table 4. Since the average document length in the Tipster collection is 806 terms (non-unique), at least 20% of the terms in the expanded query are generally irrelevant to a particular document. The results indicate that irrelevant attributes prevent all but EG from generalizing well. Comparing the performance of EG and LMS adds evidence to the Kivinen-Warmuth hypothesis that EG yields a smaller *loss* than LMS, given many irrelevant attributes. Juxtaposing the results of the 50-term and 1000-term-expansion experiments suggests that using a statistical filter for selecting the top few terms is better than expanding the query by many terms and having the learning algorithm perform term selection.

## 5   Conclusion

The experiment results presented here provide evidence that single-neuron learning algorithms can be used to improve retrieval performance in IR systems. Based on performance metrics that test the quality of a classification process and a document ranking process, the weights produced by EG were consistently better than previously available methods.

| Query type | Potential LBA | Actual LBA |
|---|---|---|
| NL | 91.9 | 88.6 |
| EXP | 95.5 | 92.0 |
| ROC | 96.7 | 94.0 |
| LMS | 92.6 | 89.8 |
| EG | 97.1 | 95.1 |

Table 3: Threshold Analysis: Query expansion by 50 terms.

| Query type | RP | LBA |
|---|---|---|
| NL | 22.0 | 88.6 |
| EXP | 14.4 | 76.5 |
| ROC | 19.7 | 82.5 |
| LMS | 20.4 | 86.7 |
| EG | 35.0 | 93.2 |

Table 4: Query expansion by 1000 terms.

## Footnotes

[1] R-Precision (RP) and Lower Bound Accuracy (LBA) performance values are normalized to a 0-100 scale. Values are reported for: NL = original natural language query; EXP = expanded query with weights set to 1.0; ROC = expanded query with weights based on Rocchio assignment; LMS = expanded query with weights based on LMS learning; and EG= expanded query with weights based on EG learning.

[2] Recent experiments using the optimization algorithm DFO (presented in [7]) suggest that certain parameter settings make it competitive with EG.

# References

[1] B. Widrow and M. Hoff, "Adaptive switching circuits", In 1960 IRE WESCON Convention Record, pp. 96-104, New York, 1960.

[2] J. Kivinen, Manfred Wartmuth, "Exponentiated Gradient Versus Gradient Descent for Linear Predictors", UCSC Tech report: UCSC-CRL-94-16, June 21, 1994.

[3] D. Lewis, R. Schapire, J. Callan, and R. Papka, "Training Algorithms for Linear Text Classifiers", Proceeding of SIGIR 1996.

[4] B.S. Wittner and J.S. Denker, "Strategies for Teaching Layered Networks Classification Tasks", NIPS proceedings, 1987.

[5] G. Salton, "Relevance Feedback and optimization of retrieval effectiveness. In The Smart system - experiments in automatic document processing", 324-336. Englewood Cliffs, NJ: Prentice Hall Inc., 1971.

[6] J.J. Rocchio, "Relevance Feedback in Information Retrieval in The Smart System - Experiments in Automatic document processing", 313-323. Englewood Cliffs, NJ: Prentice Hall Inc., 1971.

[7] C. Buckley and G. Salton, "Optimization of Relevance Feedback Weights", Proceeding of SIGIR 95 Seattle WA, 1995.

[8] J. Allan, L. Ballesteros, J. Callan, W.B. Croft, and Z. Lu, "Recent Experiments with Inquery", TREC-4 Proceedings , 1995.

[9] M. Porter, "An Algorithm for Suffix Stripping", Program, Vol 14(3), pp. 130-137, 1980.

[10] D. Harman, Proceedings of Text REtrievl Conferences (TREC), 1993-5.

[11] S.E. Robertson, W. Walker, S. Jones, M.M. Hancock-Beaulieu, and M.Gatford, "Okapi at TREC-3", TREC-3 Proceedings, 1994.

[12] G. Salton, *Automatic Text Processing*, Addison-Wesley Publishing Co, Massachusetts, 1989.

[13] S.I. Gallant, "Optimal Linear Discrimants", Proceedings of International Conference on Pattern Recognition, 1986.

[14] B.T. Bartell, "Optimizing Ranking Functions: A Connectionist Approach to Adaptive Information Retrieval", Ph.D. Theis, UCSD 1994.
